# VLSI Implementation of Cortical Visual Motion Detection Using an Analog Neural Computer

**Ralph Etienne-Cummings**
Electrical Engineering,
Southern Illinois University,
Carbondale, IL 62901

**Jan Van der Spiegel**
The Moore School,
University of Pennsylvania,
Philadelphia, PA 19104

**Naomi Takahashi**
The Moore School,
University of Pennsylvania,
Philadelphia, PA 19104

**Alyssa Apsel**
Electrical Engineering,
California Inst. Technology,
Pasadena, CA 91125

**Paul Mueller**
Corticon Inc.,
3624 Market Str,
Philadelphia, PA 19104

## Abstract

Two dimensional image motion detection neural networks have been implemented using a general purpose analog neural computer. The neural circuits perform spatiotemporal feature extraction based on the cortical motion detection model of Adelson and Bergen. The neural computer provides the neurons, synapses and synaptic time-constants required to realize the model in VLSI hardware. Results show that visual motion estimation can be implemented with simple sum-and-threshold neural hardware with temporal computational capabilities. The neural circuits compute general 2D visual motion in real-time.

## 1 INTRODUCTION

Visual motion estimation is an area where spatiotemporal computation is of fundamental importance. Each distinct motion vector traces a unique locus in the space-time domain. Hence, the problem of visual motion estimation reduces to a feature extraction task, with each feature extractor tuned to a particular motion vector. Since neural networks are particularly efficient feature extractors, they can be used to implement these visual motion estimators. Such neural circuits have been recorded in area MT of macaque monkeys, where cells are sensitive and selective to 2D velocity (Maunsell and Van Essen, 1983).

In this paper, a hardware implementation of 2D visual motion estimation with spatiotemporal feature extractors is presented. A silicon retina with parallel, continuous time edge detection capabilities is the front-end of the system. Motion detection neural networks are implemented on a general purpose analog neural computer which is composed of programmable analog neurons, synapses, axon/dendrites and synaptic time-

constants (Van der Spiegel *et al.*, 1994). The additional computational freedom introduced by the synaptic time-constants, which are unique to this neural computer, is required to realize the spatiotemporal motion estimators. The motion detection neural circuits are based on the early 1D model of Adelson and Bergen and recent 2D models of David Heeger (Adelson and Bergen, 1985; Heeger *et al.*, 1996). However, since the neurons only computed delayed weighted sum-and-threshold functions, the models must be modified. The original models require division for intensity normalization and a quadratic non-linearity to extract spatiotemporal energy. In our model, normalization is performed by the silicon retina with a large contrast sensitivity (all edges are normalized to the same output), and rectification replaces the quadratic non-linearity. Despite these modifications, we show that the model works correctly. The visual motion vector is implicitly coded as a distribution of neural activity.

Due to its computational complexity, this method of image motion estimation has not been attempted in discrete or VLSI hardware. The general purpose analog neural computer offers a unique avenue for implementing and investigating this method of visual motion estimation. The analysis, implementation and performance of spatiotemporal visual motion estimators are discussed.

## 2  SPATIOTEMPORAL FEATURE EXTRACTION

The technique of estimating motion with spatiotemporal feature extraction was proposed by Adelson and Bergen in 1985 (Adelson and Bergen, 1985). It emerged out of the observation that a point moving with constant velocity traces a line in the space-time domain, shown in figure 1a. The slope of the line is proportional to the velocity of the point. Hence, the velocity is represented as the orientation of the line. Spatiotemporal orientation detection units, similar to those proposed by Hubel and Wiesel for spatial orientation detection, can be used for detecting motion (Hubel and Wiesel, 1962). In the frequency domain, the motion of the point is also a line where the slope of the line is the velocity of the point. Hence orientation detection filters, shown as circles in figure 1b, can be used to measure the motion of the point relative to their tuned velocity. A population of these tuned filters, figure 1c, can be used to measure general image motion.

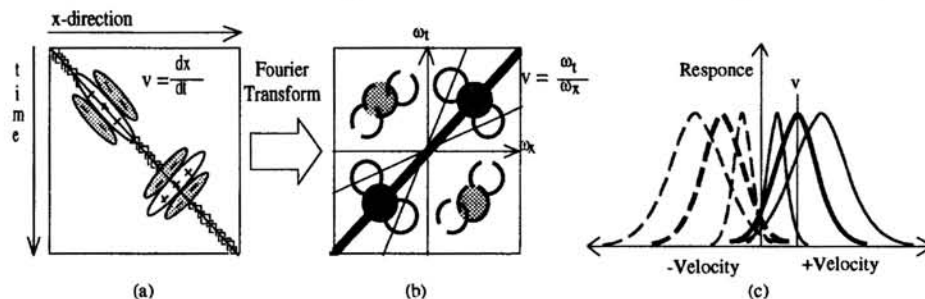

Figure 1: (a) 1D Motion as Orientation in the Space-Time Domain.
(b) and (c) Motion detection with Oriented Spatiotemporal Filters.

If the point exhibits 2D motion, the problem is substantially more complicated, as observed by David Heeger (1987). A point executing 2D motion spans a plane in the frequency domain. The spatiotemporal orientation filter tuned to this motion must also span a plane (Heeger *et al.*, 1987, 1996). Figure 2a shows a filter tuned to 2D motion. Unfortunately, this torus shaped filter is difficult to realize without special mathematical tools. Furthermore, to create a general set of filters for measuring general 2D motion, the filters must cover all the spatiotemporal frequencies and all the possible velocities of the stimuli. The latter requirement is particularly difficult to obtain since there are two degrees of freedom ($v_x$, $v_y$) to cover.

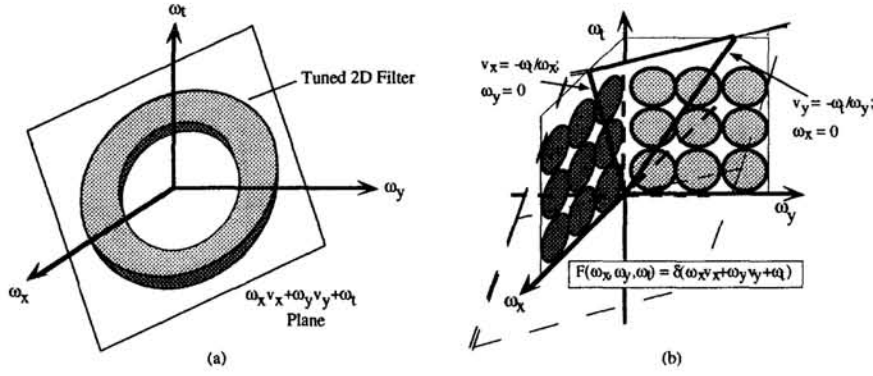

Figure 2: (a) 2D Motion Detection with 2D Oriented Spatiotemporal Filters. (b) General 2D Motion Detection with 2 Sets of 1D Filters.

To circumvent these problems, our model decomposes the image into two orthogonal images, where the perpendicular spatial variation within the receptive field of the filters are eliminated using spatial smoothing. Subsequently, 1D spatiotemporal motion detection is used on each image to measure the velocity of the stimuli. This technique places the motion detection filters, shown as the circles in figure 2b, only in the $\omega_x$-$\omega_t$ and $\omega_y$-$\omega_t$ planes to extract 2D motion, thereby drastically reducing the complexity of the 2D motion detection model from $O(n^2)$ to $O(2n)$.

## 2.1  CONSTRUCTING THE SPATIOTEMPORAL MOTION FILTERS

The filter tuned to a velocity $v_{ox}$ ($v_{0y}$) is centered at $\omega_{ox}$ ($\omega_{oy}$) and $\omega_{ot}$ where $v_{ox} = \omega_{ot}/\omega_{ox}$ ($v_{0y} = \omega_{ot}/\omega_{0y}$). To create the filters, quadrature pairs (i.e. odd and even pairs) of spatial and temporal band-pass filters centered at the appropriate spatiotemporal frequencies are summed and differenced (Adelson and Bergen, 1985). The $\pi/2$ phase relationship between the filters allows them to be combined such that they cancel in opposite quadrants, leaving the desired oriented filter, as shown in figure 3a. Equation 1 shows examples of quadrature pairs of spatial and temporal filters implemented. The coefficients of the filters balance the area under their positive and negative lobes. The spatial filters in equation 1 have a 5 x 5 receptive field, where the sampling interval is determined by the silicon retina. Figure 3b shows a contour plot of an oriented filter ($\alpha$=11 rads/s, $\delta_2$=2$\delta_1$=40$\alpha$).

$$S(even) = [0.5 - 0.32 Cos(\omega_x) - 0.18 Cos(2\omega_x)] \qquad (a)$$

$$S(odd) = [-0.66 j Sin(\omega_x) - 0.32 j Sin(2\omega_x)] \qquad (b)$$

$$T(even) = \frac{-\omega_t^2 \delta_2}{(j\omega_t + \alpha)(j\omega_t + \delta_1)(j\omega_t + \delta_2)}; \alpha << \delta_1 \approx \delta_2 \qquad (c)$$

$$T(odd) = \frac{j\omega_t \delta_1 \delta_2}{(j\omega_t + \alpha)(j\omega_t + \delta_1)(j\omega_t + \delta_2)}; \alpha << \delta_1 \approx \delta_2 \qquad (d)$$

(1)

$$Left\ Motion = S(e)T(e) - S(o)T(o)\ or\ S(e)T(o) - S(o)T(e) \qquad (e)$$

$$Right\ Motion = S(e)T(e) + S(o)T(o)\ or\ S(e)T(o) + S(o)T(e) \qquad (f)$$

To cover a wide range of velocity and stimuli, multiple filters are constructed with various velocity, spatial and temporal frequency selectivity. Nine filters are chosen per dimension to mosaic the $\omega_x$-$\omega_t$ and $\omega_y$-$\omega_t$ planes as in figure 2b. The velocity of a stimulus is given by the weighted average of the tuned velocity of the filters, where the weights are the magnitudes of each filter's response. All computations for 2D motion detection based on cortical models have been realized in hardware using a large scale general purpose analog neural computer.

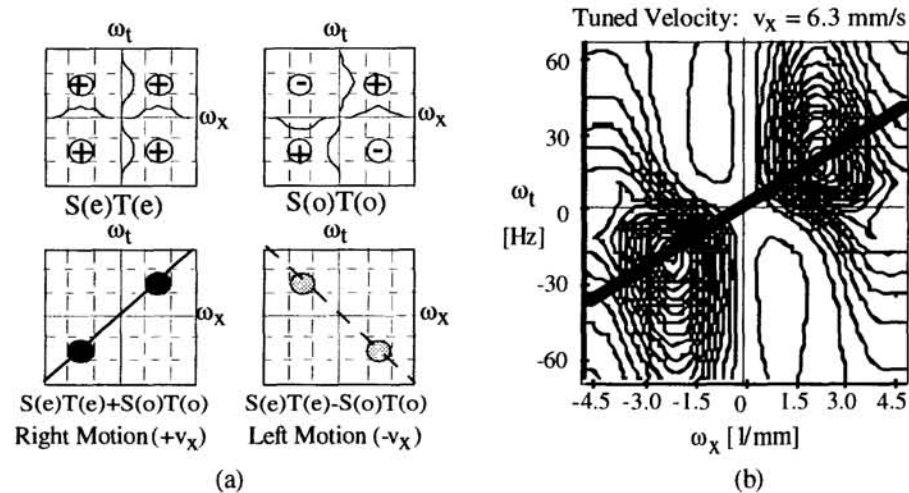

Figure 3: (a) Constructing Oriented Spatiotemporal Filters. (b) Contour Plot of One of the Filters Implemented.

## 3  HARDWARE IMPLEMENTATION

### 3.1  GENERAL PURPOSE ANALOG NEURAL COMPUTER

The computer is intended for fast prototyping of neural network based applications. It offers the flexibility of programming combined with the real-time performance of a hardware system (Mueller, 1995). It is modeled after the biological nervous system, i.e. the cerebral cortex, and consists of electronic analogs of neurons, synapses, synaptic time constants and axon/dendrites. The hardware modules capture the functional and computational aspects of the biological counterparts. The main features of the system are: configurable interconnection architecture, programmable neural elements, modular and expandable architecture, and spatiotemporal processing. These features make the network ideal to implement a wide range of network architectures and applications.

The system, shown in part in figure 4, is constructed from three types of modules (chips): (1) neurons, (2) synapses and (3) synaptic time constants and axon/dendrites. The neurons have a piece-wise linear transfer function with programmable (8bit) threshold and minimum output at threshold. The synapses are implemented as a programmable resistance whose values are variable (8 bit) over a logarithmic range between 5KOhm and 10Mohm. The time constant, realized with a load-compensated transconductance amplifier, is selectable between 0.5ms and 1s with a 5 bit resolution. The axon/dendrites are implemented with an analog cross-point switch matrix. The neural computer has a total of 1024 neurons, distributed over 64 neuron modules, with 96 synaptic inputs per neuron, a total of 98,304 synapses, 6,656 time constants and 196,608 cross point switches. Up to 3,072 parallel buffered analog inputs/outputs and a neuron output analog mulitplexer are available. A graphical interface software, which runs on the host computer, allows the user to symbolically and physically configure the network and display its behavior (Donham, 1994). Once a particular network has been loaded, the neural network runs independently of the digital host and operates in a fully analog, parallel and continuous time fashion.

### 3.2  NEURAL IMPLEMENTATION OF SPATIOTEMPORAL FILTERS

The output of the silicon retina, which transforms a gray scale image into a binary image of edges, is presented to the neural computer to implement the oriented spatiotemporal filters. The first and second derivatives of Gaussian functions are chosen to implement the odd and even spatial filters, respectively. They are realized by feeding the outputs of

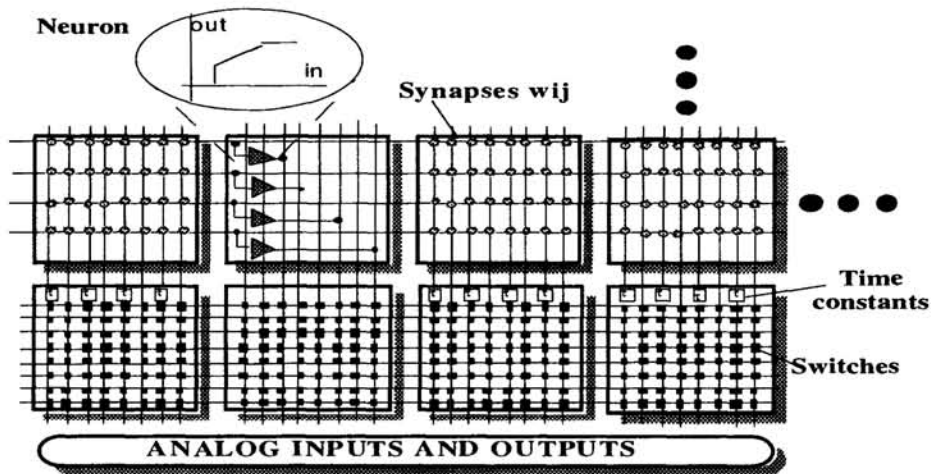

Figure 4: Block Diagram of the Overall Neural Network Architecture.

the retina, with appropriate weights, into a layer of neurons. Three parallel channels with varying spatial scales are implemented for each dimension. The output of the even (odd) spatial filter is subsequently fed to three parallel even (odd) temporal filters, which also have varying temporal tuning. Hence, three *non-oriented* pairs of spatiotemporal filters are realized for each channel. Six oriented filters are realized by summing and differencing the non-oriented pairs. The oriented filters are rectified, and lateral inhibition is used to accentuate the higher response. Figure 4 shows a schematic of the neural circuitry used to implement the orientation selective filters.

The image layer of the network in figure 5 is the direct, parallel output of the silicon retina. A 7 x 7 pixel array from the retina is decomposed into 2, 1 x 7 orthogonal linear images, and the nine motion detection filters are implemented per image. The total number of neurons used to implement this network is 152, the number of synapse is 548 and the number of time-constants is 108. The time-constant values ranges from 0.75 ms to 375 ms. After the networks have been programmed into the VLSI chips of the neural computer, the system operates in full parallel and continuous time analog mode. Consequently, this system realizes a silicon model for biological visual image motion measurement, starting from the retina to the visual cortex.

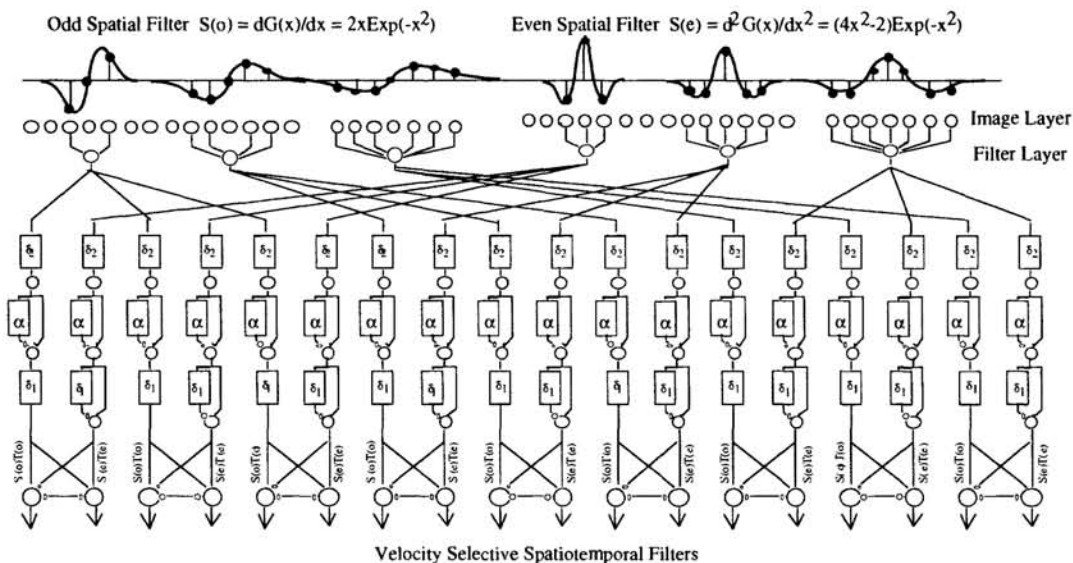

Figure 5: Neural Network Implementation of the Oriented Spatiotemporal Filters.

## 4   RESULTS

The response of the spatiotemporal filters implemented with the neural computer are shown in figure 6. The figure is obtained by sampling the output of the neurons at 1MHz using the on-chip analog multiplexers. In figure 6a, the impulse response of the spatial filters are shown as a point moves across their receptive field. Figure 6b shows the outputs of the even and odd temporal filters for the moving point. At the output of the filters, the even and odd signals from the spatial filters are no longer out of phase. This transformation yields to constructive or destructive interference when they are summed and differenced. When the point move in opposite direction, the odd filters changes such that the output of the temporal filters become 180° out of phase. Subsequent summing and differencing will have the opposite result. Figure 6c shows the output for all nine x-velocity selective filters as a point moves with positive velocity.

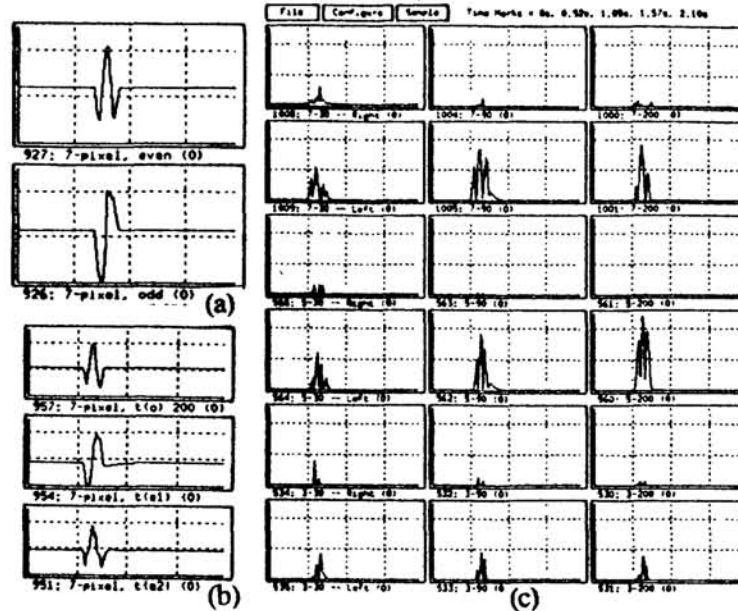

Figure 6:   Output of the Neural Circuits for a Moving Point:   (a) Spatial Filters, (b) Temporal Filters and (c) Motion Filters.

Figure 7 shows the tuning curves for the filters tuned to x-motion. The variations in the responses are due to variations in the analog components of the neural computer. Some aliasing is noticeable in the tuning curves when there is a minor peak in the opposite direction. This results from the discrete properties of the spatial filters, as seen in

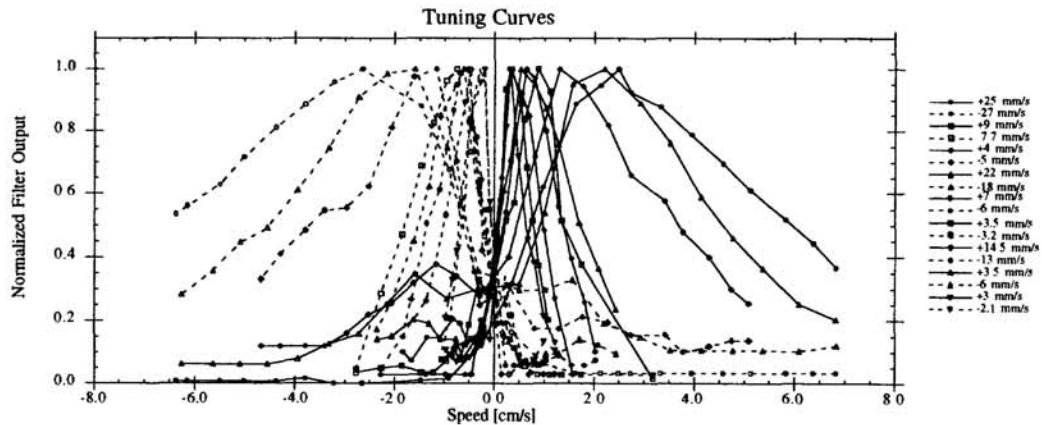

Figure 7:   Tuning Curves for the Nine X-Motion Filters.

figure 3b. Due to the lateral inhibition employed, the aliasing effects are minimal. Similar curves are obtained for the y-motion tuned filters.

For a point moving with $v_x = 8.66$ mm/s and $v_y = 5$ mm/s, the output of the motion filters are shown in Table 1. Computing a weighted average using equation 2, yields $v_{xm} = 8.4$ mm/s and $v_{ym} = 5.14$ mm/s. This result agrees with the actual motion of the point.

$$v_m = \sum_i v_i^{tuned} O_i \bigg/ \sum_i O_i \tag{2}$$

Table 1: Filter Responses for a Point Moving at 10 mm/s at 30°.

| | X Filters [Speed in mm/s] | | | | | | | | | Y Filters [Speed in mm/s] | | | | | | | | |
|---|---|---|---|---|---|---|---|---|---|---|---|---|---|---|---|---|---|---|
| Tuned Speed | 25 | 9 | 4 | 22 | 7 | 3.5 | 14.5 | 3.5 | 3 | 26 | 9.5 | 5 | 20 | 7.8 | 3.7 | 15 | 4.1 | 3.5 |
| Response | 0.52 | 0.95 | 0.57 | 0.53 | 0.9 | 0.3 | 0.75 | 0.9 | 0.31 | 0.35 | 0.675 | 0.92 | 0.3 | 0.85 | 0.9 | 0.54 | 0.9 | 0.9 |
| Tuned Speed | -27 | -7.7 | -5 | -18 | -6 | -3.2 | -13 | -6 | -2.1 | -25 | -8 | -4.1 | -21 | -7 | -4 | -14 | -5 | -2 |
| Response | 0.0 | 0.05 | 0.1 | 0.1 | 0.05 | 0.05 | 0.02 | 0.05 | 0.1 | 0.1 | 0.08 | 0.1 | 0.3 | 0.05 | 0.01 | 0.23 | 0.05 | 0.1 |

# 5 CONCLUSION

2D image motion estimation based on spatiotemporal feature extraction has been implemented in VLSI hardware using a general purpose analog neural computer. The neural circuits capitalize on the temporal processing capabilities of the neural computer. The spatiotemporal feature extraction approach is based on the 1D cortical motion detection model proposed by Adelson and Bergen, which was extended to 2D by Heeger *et al.* To reduce the complexity of the model and to allow realization with simple sum-and-threshold neurons, we further modify the 2D model by placing filters only in the $\omega_x$-$\omega_t$ and $\omega_y$-$\omega_t$ planes, and by replacing quadratic non-linearities with a rectifiers. The modifications do not affect the performance of the model. While this technique of image motion detection requires too much hardware for focal plane implementation, our results show that it is realizable when a silicon "brain," with large numbers of neurons and synaptic time constant, is available. This is very reminiscent of the biological master.

## References

E. Adelson and J. Bergen, "Spatiotemporal Energy Models for the Perception of Motion," *J. Optical Society of America,* Vol. A2, pp. 284-99, 1985

C. Donham, "Real Time Speech Recognition using a General Purpose Analog Neurocomputer," *Ph.D. Thesis,* Univ. of Pennsylvania, Dept. of Electrical Engineering, Philadelphia, PA, 1995.

D. Heeger, E. Simoncelli and J. Movshon, "Computational Models of Cortical Visual Processing," *Proc. National Academy of Science,* Vol. 92, no. 2, pp. 623, 1996

D. Heeger, "Model for the Extraction of Image Flow," *J. Optical Society of America,* Vol. 4, no. 8, pp. 1455-71, 1987

D. Hubel and T. Wiesel, "Receptive Fields, Binocular Interaction and Functional Architecture in the Cat's Visual Cortex," *J. Physiology,* Vol. 160, pp. 106-154, 1962

J. Maunsell and D. Van Essen, "Functional Properties of Neurons in Middle Temporal Visual Area of the Macaque Monkey. I. Selectivity for Stimulus Direction, Speed and Orientation," *J. Neurophysiology,* Vol. 49, no. 5, pp. 1127-47, 1983

P. Mueller, J. Van der Spiegel, D. Blackman, C. Donham and R. Etienne-Cummings, "A Programmable Analog Neural Computer with Applications to Speech Recognition," *Proc. Comp. & Info. Sci. Symp.* (CISS), J. Hopkins, May 1995.
